# Feature Densities are Required for Computing Feature Correspondences

**Subutai Ahmad**
Interval Research Corporation
1801-C Page Mill Road, Palo Alto, CA 94304
E-mail: ahmad@interval.com

## Abstract

The feature correspondence problem is a classic hurdle in visual object-recognition concerned with determining the correct mapping between the features measured from the image and the features expected by the model. In this paper we show that determining good correspondences requires information about the joint probability density over the image features. We propose "likelihood based correspondence matching" as a general principle for selecting optimal correspondences. The approach is applicable to non-rigid models, allows nonlinear perspective transformations, and can optimally deal with occlusions and missing features. Experiments with rigid and non-rigid 3D hand gesture recognition support the theory. The likelihood based techniques show almost no decrease in classification performance when compared to performance with perfect correspondence knowledge.

## 1 INTRODUCTION

The ability to deal with missing information is crucial in model-based vision systems. The feature correspondence problem is an example where the correct mapping between image features and model features is unknown at recognition time. For example, imagine a network trained to map fingertip locations to hand gestures. Given features extracted from an image, it becomes important to determine which features correspond to the thumb, to the index finger, etc. so we know which input units to clamp with which numbers. Success at the correspondence matching step

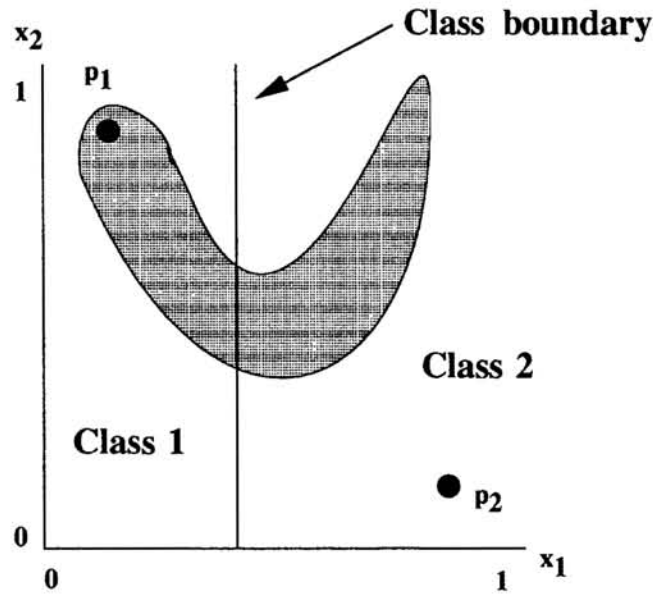

Figure 1: An example 2D feature space. Shaded regions denote high probability. Given measured values of 0.2 and 0.9, the points $p_1$ and $p_2$ denote possible instantiations but $p_1$ is much more likely.

is vital for correct classification. There has been much previous work on this topic (Connell and Brady 1987; Segen 1989; Huttenlocher and Ullman 1990; Pope and Lowe 1993) but a general solution has eluded the vision community. In this paper we propose a novel approach based on maximizing the probability of a set of models generating the given data. We show that neural networks trained to estimate the joint density between image features can be successfully used to recover the optimal correspondence. Unlike other techniques, the likelihood based approach is applicable to non-rigid models, allows perspective 3D transformations, and includes a principled method for dealing with occlusions and missing features.

## 1.1  A SIMPLE EXAMPLE

Consider the idealized example depicted in Figure 1. The distribution of features is highly non-uniform (this is typical of non-rigid objects). The classification boundary is in general completely unrelated to the feature distribution. In this case, the class (posterior) probability approaches 1 as feature $x_1$ approaches 0, and 0 as it approaches 1. Now suppose that two feature values 0.2 and 0.9 are measured from an image. The task is to decide which value gets assigned to $x_1$ and which value gets assigned to $x_2$. A common strategy is to select the correspondence which gives the maximal network output (i.e. maximal posterior probability). In this example (and in general) such a strategy will pick point $p_2$, the wrong correspondence. This is because the classifer output represents the probability of a class given a specific feature assignment and specific values. The correspondence problem however, is something completely different: it deals with the probability of getting the feature assignments and values in the first place.

## 2 LIKELIHOOD BASED CORRESPONDENCE MATCHING

We can formalize the intuitive arguments in the previous section. Let $C$ denote the set of classes under consideration. Let $X$ denote the list of features measured from the image with correspondences unknown. Let $A$ be the set of assignments of the measured values to the model features. Each assignment $a \in A$ reflects a particular choice of feature correspondences. We consider two different problems: the task of choosing the best assignment $a$ and the task of classifying the object given $X$.

Selecting the best correspondence is equivalent to selecting the permutation that maximizes $p(a|X,C)$. This can be re-written as:

$$p(a|X,C) = \frac{p(X|a,C)p(a|C)}{p(X|C)} \tag{1}$$

$p(X|C)$ is a normalization factor that is constant across all $a$ and can be ignored. Let $\mathbf{x}_a$ denote a specific feature vector constructed by applying permutation $a$ to $X$. Then (1) is equivalent to maximizing:

$$p(a|X,C) = p(\mathbf{x}_a|C)p(a|C) \tag{2}$$

$p(a|C)$ denotes our prior knowledge about possible correspondences. (For example the knowledge that edge features cannot be matched to color features.) When no prior knowledge is available this term is constant. We denote the assignment that maximizes (2) the *maximum likelihood correspondence match*. Such a correspondence maximizes the probability that a set of visual models generated a given set of image features and will be the optimal correspondence in a Bayesian sense.

### 2.1 CLASSIFICATION

In addition to computing correspondences, we would like to classify a model from the measured image features, i.e. compute $p(C_i|X,C)$. The maximal-output based solution is equivalent to selecting the class $C_i$ that maximizes $p(C_i|\mathbf{x}_a,C)$ over all assignments $a$ and all classes $C_i$. It is easy to see that the optimal strategy is actually to compute the following weighted estimate over all candidate assignments:

$$p(C_i|X,C) = \frac{\sum_a p(C_i|X,a,C)p(X|a,C)p(a|C)}{p(X|C)} \tag{3}$$

Classification based on (3) is equivalent to selecting the class that maximizes:

$$\sum_a p(C_i|\mathbf{x}_a,C)p(\mathbf{x}_a|C)p(a|C) \tag{4}$$

Note that the network output based solution represents quite a degraded estimate of (4). It does not consider the input density nor perform a weighting over possible

correspondences. A reasonable approximation is to select the maximum likelihood correspondence according to (2) and then use this feature vector in the classification network. This is suboptimal since the weighting is not done but in our experience it yields results that are very close to those obtained with (4).

# 3    COMPUTING CORRESPONDENCES WITH GBF NETWORKS

In order to compute (2) and (4) we consider networks of normalized Gaussian basis functions (GBF networks). The $i$'th output unit is computed as:

$$y_i(\mathbf{x}) = \frac{\sum_j w_{ij} b_j(\mathbf{x})}{\sum_j b_j(\mathbf{x})} \tag{5}$$

with:

$$b_j(\mathbf{x}) = \pi_j n(\mathbf{x}; \mu_{\mathbf{j}}, \sigma_{\mathbf{j}}) = \frac{\pi_j}{(2\pi)^{d/2} \prod_{k=1}^d \sigma_{kj}} \exp[-\sum_i \frac{(x_i - \mu_{ji})^2}{2\sigma_{ji}^2}] \tag{6}$$

Here each basis function $j$ is characterized by a mean vector $\mu_{\mathbf{j}}$ and by $\sigma_{\mathbf{j}}$, a vector representing the diagonal covariance matrix. $w_{ji}$ represents the weight from the $j$'th Gaussian to the $i$'th output. $\pi_j$ is a weight attached to each basis function.

Such networks have been popular recently and have proven to be useful in a number of applications (e.g. (Roscheisen et al. 1992; Poggio and Edelman 1990). For our current purpose, these networks have a number of advantages. Under certain training regimes such as EM or "soft clustering" (Dempster et al. 1977; Nowlan 1990) or an approximation such as K-Means (Neal and Hinton 1993), the basis functions adapt to represent local probability densities. In particular $p(\mathbf{x}_a|C) \approx \sum_j b_j(\mathbf{x}_a)$. If standard error gradient training is used to set the weights $w_{ij}$ then $y_i(\mathbf{x}_a) \approx p(C_i|\mathbf{x}_a, C)$ Thus both (2) and (4) can be easilty computed.(Ahmad and Tresp 1993) showed that such networks can effectively learn feature density information for complex visual problems. (Poggio and Edelman 1990) have also shown that similar networks (with a different training regime) can learn to approximate the complex mappings that arise in 3D recognition.

## 3.1    OPTIMAL CORRESPONDENCE MATCHING WITH OCCLUSION

An additional advantage of GBF networks trained in this way is that it is possible to obtain closed form solutions to the optimal classifier in the presence of missing or noisy features. It is also possible to correctly compute the probability of feature vectors containing missing dimensions. The solution consists of projecting each Gaussian onto the non-missing dimensions and evaluating the resulting network. Note that it is incorrect to simply substitute zero or any other single value for the missing dimensions. (For lack of space we refer the reader to (Ahmad and Tresp

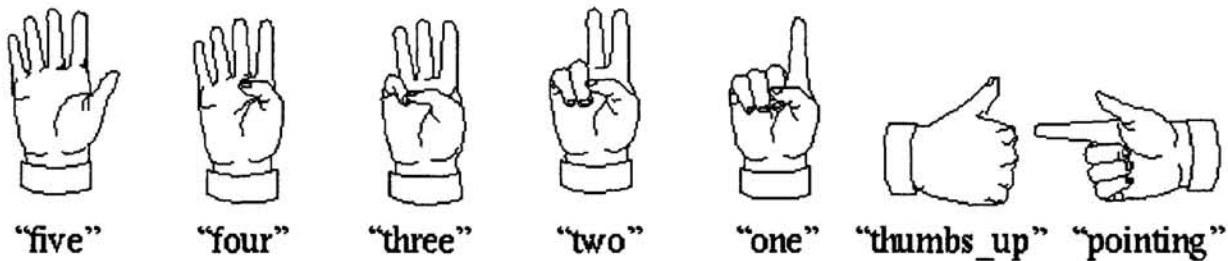

"five"    "four"    "three"    "two"    "one"    "thumbs_up"  "pointing"

Figure 2: Classifiers were trained to recognize these 7 gestures. a $3D$ computer model of the hand is used to generate images of the hand in various poses. For each training example, we randomly choose a $3D$ orientation and depth, compute the $3D$ positions of the fingertips and project them onto $2D$. There were 5 features yielding a $10D$ input space.

1993) for further details.) Thus likelihood based approaches using GBF networks can simultaneously optimally deal with occlusions and the correspondence problem.

## 4    EXPERIMENTAL RESULTS

We have used the task of $3D$ gesture recognition to compare likelihood based methods to the network output based technique. (Figure 2 describes the task.) We considered both rigid and non-rigid gesture recognition tasks. We used a GBF network with 10 inputs, 1050 basis functions and 7 output units. For comparision we also trained a standard backpropagation network (BP) with 60 hidden units on the task. For this task we assume that during training all feature correspondences are known and that during training no feature values are noisy or missing. Classification performance with full correspondence information on an independent test set is about 92% for the GBF network and 93% for the BP network. (For other results see (Williams et al. 1993) who have also used the rigid version of this task as a benchmark.)

### 4.1    EXPERIMENTS WITH RIGID HAND POSES

Table 1 plots the ability of the various methods to select the correct correspondence. Random patterns were selected from the test set and all $5! = 120$ possible combinations were tried. MLCM denotes the percentage of times the maximum likelihood method (equation (2)) selected the correct feature correspondence. GBF-M and BP-M denotes how often the maximal output method chooses the correct correspondence using GBF nets and BP. "Random" denotes the percentage if correspondences are chosen randomly. The substantially better performance of MLCM suggests that, at least for this task, density information is crucial. It is also interesting to examine the errors made by MLCM. A common error is to switch the features for the pinky and the adjacent finger for gestures "one", "two", "thumbs-up" and "pointing". These two fingertips often project very close to one another in many poses; such a mistake usually do not affect subsequent classification.

| Selection Method | Percentage Correct |
|---|---|
| Random | 1.2% |
| GBF-M | 8.8% |
| BP-M | 10.3% |
| MLCM | 62.0% |

Table 1: Percentage of correspondences selected correctly.

| Classifier | Classification Performance |
|---|---|
| BP-Random | 28.0% |
| BP-Max | 39.2% |
| GBF-Max | 47.3% |
| GBF-WLC | 86.2% |
| GBF-Known | 91.8% |

Table 2: Classification without correspondence information.

Table 2 shows classification performance when the correspondence is unknown. GBF-WLC denotes weighted likelihood classification using GBF networks to compute the feature densities and the posterior probabilities. Performance with the output based techniques are denoted with GBF-M and BP-M. BP-R denotes performance with random correspondences using the back propagation network. GBF-known plots the performance of the GBF network when all correspondences are known. The results are quite encouraging in that performance is only slightly degraded with WLC even though there is substantially less information present when correspondences are unknown. Although not shown, results with MLCM (i.e. not doing the weighting step but just choosing the correspondence with highest probability) are about 1% less than WLC. This supports the theory that many of the errors of MLCM in Table 1 are inconsequential.

### 4.1.1   Missing Features and No Correspondences

Figure 3 shows error as a function of the number of missing dimensions. (The missing dimensions were randomly selected from the test set.) Figure 3 plots the average number of classes that are assigned higher probability than the correct class. The network output method and weighted likelihood classification is compared to the case where all correspondences are known. In all cases the basis functions were projected onto the non-missing dimensions to approximate the Bayes-optimal condition. As before, the likelihood based method outperforms the output based method. Surprisingly, even with 4 of the 10 dimensions missing and with correspondences unknown, WLC assigns highest probability to the correct class on average (performance score < 1.0).

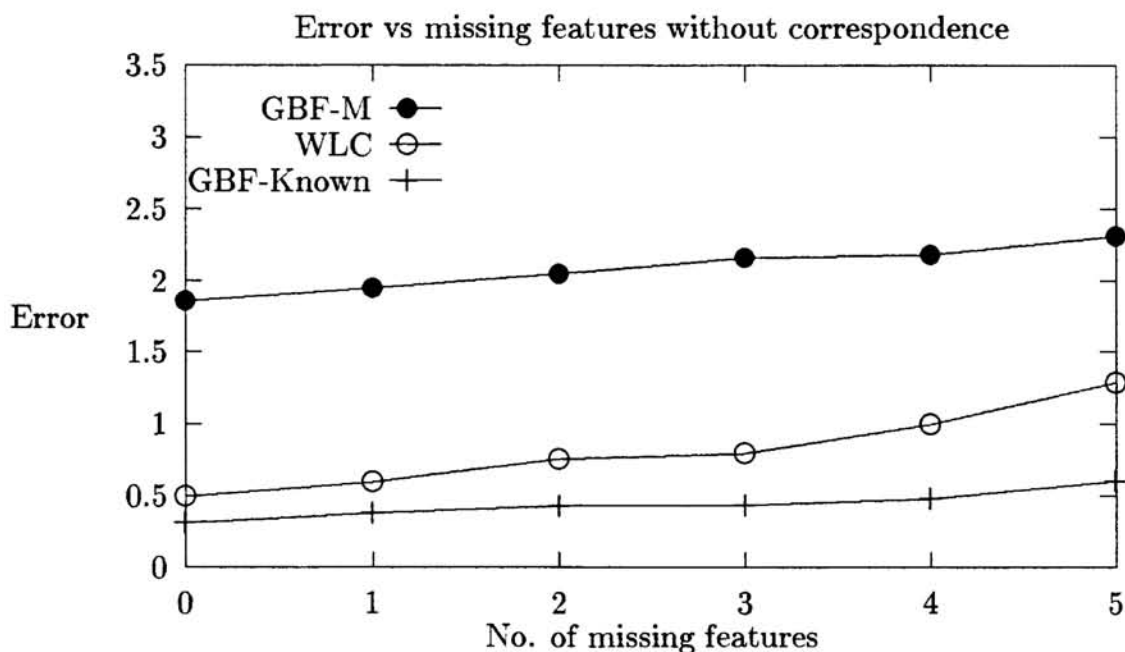

Figure 3: Error with missing features when no correspondence information is present. The $y$-axis denotes the average number of classes that are assigned higher probability than the correct class.

## 4.2  EXPERIMENTS WITH NON-RIGID HAND POSES

In the previous experiments the hand configuration for each gesture remained rigid. Correspondence selection with non-rigid gestures was also tried out. As before a training set consisting of examples of each gesture was constructed. However, in this case, for each sample, a random perturbation (within 20 degrees) was added to each finger joint. The orientation of each sample was allowed to randomly vary by 45 degrees around the x, y, and z axes. When viewed on a screen the samples give the appearance of a hand wiggling around. Surprisingly, GBF networks with 210 hidden units consistently selected the correct correspondences with a performance of 94.9%. (The performance is actually better than the rigid case. This is because in this training set all possible 3D orientations were not allowed.)

## 5  DISCUSSION

We have shown that estimates of joint feature densities can be used to successfully deal with lack of correspondence information even when some input features are missing. We have dealt mainly with the rather severe case where no prior information about correspondences is available. In this particular case to get the optimal correspondece, all $n!$ possibilities must be considered. However this is usually not necessary. Useful techniques exist for reducing the number of possible correspondences. For example, (Huttenlocher and Ullman 1990) have argued that three fea-

ture correspondences are enough to constrain the pose of rigid objects. In this case only $O(n^3)$ matches need to be tested. In addition features usually fall into incompatible sets (e.g. edge features, corner features, etc.) further reducing the number of potential matches. Finally, with image sequences one can use correspondence information from the previous frame to constrain the set of correspondences in the current frame. Whatever the situation, a likelihood based approach is a principled method for evaluating the set of available matches.

## Acknowledgements

Much of this research was conducted at Siemens Central Research in Munich, Germany. I would like to thank Volker Tresp at Siemens for many interesting discussions and Brigitte Wirtz for providing the hand model.

## References

Ahmad, S. and V. Tresp (1993). Some solutions to the missing feature problem in vision. In S. Hanson, J. Cowan, and C. Giles (Eds.), *Advances in Neural Information Processing Systems 5*, pp. 393 400. Morgan Kaufmann Publishers.

Connell, J. and M. Brady (1987). Generating and generalizing models of visual objects. *Artificial Intelligence 31*, 159 183.

Dempster, A., N. Laird, and D. Rubin (1977). Maximum-likelihood from incomplete data via the EM algorithm. *J. Royal Statistical Soc. Ser. B 39*, 1 38.

Huttenlocher, D. and S. Ullman (1990). Recognizing solid objects by alignment with an image. *International Journal of Computer Vision 5*(2), 195 212.

Neal, R. and G. Hinton (1993). A new view of the EM algorithm that justifies incremental and other variants. *Biometrika, submitted*.

Nowlan, S. (1990). Maximum likelihood competitive learning. In D. Touretzky (Ed.), *Advances in Neural Information Processing Systems 2*, pp. 574 582. San Mateo, CA: Morgan Kaufmann Publishers.

Poggio, T. and S. Edelman (1990). A network that learns to recognize three-dimensional objects. *Nature 343*(6225), 1 3.

Pope, A. and D. Lowe (1993, May). Learning object recognition models from images. In *Fourth International Conference on Computer Vision*, Berlin. IEEE Computer Society Press.

Roscheisen, M., R. Hofmann, and V. Tresp (1992). Neural control for rolling mills: Incorporating domain theories to overcome data deficiency. In M. J., H. S.J., and L. R. (Eds.), *Advances in Neural Information Processing Systems 4*. Morgan Kaufman.

Segen, J. (1989). Model learning and recognition of nonrigid objects. In *IEEE Computer Society Conference on Computer Vision and Pattern Recognition*, San Diego, CA.

Williams, C. K., R. S. Zemel, and M. C. Mozer (1993). Unsupervised learning of object models. In *AAAI Fall 1993 Symposium on Machine Learning in Computer Vision*, pp. 20 24. Proceedings available as AAAI Tech Report FSS-93-04.